# NeuroScale: Novel Topographic Feature Extraction using RBF Networks

**David Lowe**
D.Lowe@aston.ac.uk

**Michael E. Tipping**
M.E.Tipping@aston.ac.uk

Neural Computing Research Group
Aston University, Aston Triangle, Birmingam B4 7ET, UK
http://www.ncrg.aston.ac.uk/

## Abstract

Dimension-reducing feature extraction neural network techniques which also preserve neighbourhood relationships in data have traditionally been the exclusive domain of Kohonen self organising maps. Recently, we introduced a novel dimension-reducing feature extraction process, which is also topographic, based upon a Radial Basis Function architecture. It has been observed that the generalisation performance of the system is broadly insensitive to model order complexity and other smoothing factors such as the kernel widths, contrary to intuition derived from supervised neural network models. In this paper we provide an effective demonstration of this property and give a theoretical justification for the apparent 'self-regularising' behaviour of the 'NEUROSCALE' architecture.

## 1 'NeuroScale': A Feed-forward Neural Network Topographic Transformation

Recently an important class of topographic neural network based feature extraction approaches, which can be related to the traditional statistical methods of Sammon Mappings (Sammon, 1969) and Multidimensional Scaling (Kruskal, 1964), have been introduced (Mao and Jain, 1995; Lowe, 1993; Webb, 1995; Lowe and Tipping, 1996). These novel alternatives to Kohonen-like approaches for topographic feature extraction possess several interesting properties. For instance, the NEUROSCALE architecture has the empirically observed property that the generalisation perfor-

mance does not seem to depend critically on model order complexity, contrary to intuition based upon knowledge of its supervised counterparts. This paper presents evidence for their 'self-regularising' behaviour and provides an explanation in terms of the curvature of the trained models.

We now provide a brief introduction to the NEUROSCALE philosophy of nonlinear topographic feature extraction. Further details may be found in (Lowe, 1993; Lowe and Tipping, 1996). We seek a dimension-reducing, *topographic* transformation of data for the purposes of visualisation and analysis. By 'topographic', we imply that the geometric structure of the data be optimally preserved in the transformation, and the embodiment of this constraint is that the inter-point distances in the feature space should correspond as closely as possible to those distances in the data space. The implementation of this principle by a neural network is very simple. A Radial Basis Function (RBF) neural network is utilised to predict the coordinates of the data point in the transformed feature space. The locations of the feature points are indirectly determined by adjusting the weights of the network. The transformation is determined by optimising the network parameters in order to minimise a suitable error measure that embodies the topographic principle.

The specific details of this alternative approach are as follows. Given an $m$-dimensional input space of $N$ data points $\mathbf{x}_q$, an $n$-dimensional feature space of points $\mathbf{y}_q$ is generated such that the relative positions of the feature space points minimise the error, or 'STRESS', term:

$$E = \sum_p \sum_{q>p}^{N} (d_{qp}^* - d_{qp})^2, \tag{1}$$

where the $d_{qp}^*$ are the inter-point Euclidean distances in the data space: $d_{qp}^* = \sqrt{(\mathbf{x}_q - \mathbf{x}_p)^{\mathsf{T}}(\mathbf{x}_q - \mathbf{x}_p)}$, and the $d_{qp}$ are the corresponding distances in the feature space: $d_{qp} = \sqrt{(\mathbf{y}_q - \mathbf{y}_p)^{\mathsf{T}}(\mathbf{y}_q - \mathbf{y}_p)}$.

The points $\mathbf{y}$ are generated by the RBF, given the data points as input. That is, $\mathbf{y}_q = \mathbf{f}(\mathbf{x}_q; \mathbf{W})$, where $\mathbf{f}$ is the nonlinear transformation effected by the RBF with parameters (weights and any kernel smoothing factors) $\mathbf{W}$. The distances in the feature space may thus be given by $d_{qp} = \| \mathbf{f}(\mathbf{x}_q) - \mathbf{f}(\mathbf{x}_p) \|$ and so more explicitly by

$$d_{qp}^2 = \sum_{l=1}^{n} \left( \sum_k w_{lk} \left[ \phi_k(\| \mathbf{x}_q - \boldsymbol{\mu}_k \|) - \phi_k(\| \mathbf{x}_p - \boldsymbol{\mu}_k \|) \right] \right)^2, \tag{2}$$

where $\phi_k()$ are the basis functions, $\boldsymbol{\mu}_k$ are the centres of those functions, which are fixed, and $w_{lk}$ are the weights from the basis functions to the output.

The topographic nature of the transformation is imposed by the STRESS term which attempts to match the inter-point Euclidean distances in the feature space with those in the input space. This mapping is *relatively supervised* because there is no specific target for each $\mathbf{y}_q$; only a relative measure of target separation between each $\mathbf{y}_q, \mathbf{y}_p$ pair is provided. In this form it does not take account of any additional information (for example, class labels) that might be associated with the data points, but is determined strictly by their spatial distribution. However, the approach may be extended to incorporate the use of extra 'subjective' information which may be

used to influence the transformation and permits the extraction of 'enhanced', more informative, feature spaces (Lowe and Tipping, 1996).

Combining equations (1) and (2) and differentiating with respect to the weights in the network allows the partial derivatives of the STRESS $\partial E/\partial w_{lk}$ to be derived for each pattern pair. These may be accumulated over the entire pattern set and the weights adjusted by an iterative procedure to minimise the STRESS term $E$. Note that the objective function for the RBF is no longer quadratic, and so a standard analytic matrix-inversion method for fixing the final layer weights cannot be employed.

We refer to this overall procedure as 'NEUROSCALE'. Although any universal approximator may be exploited within NEUROSCALE, using a Radial Basis Function network allows more theoretical analysis of the resulting behaviour, despite the fact that we have lost the usual linearity advantages of the RBF because of the STRESS measure. A schematic of the NEUROSCALE model is given in figure 1, and illustrates the rôle of the RBF in transforming the data space to the feature space.

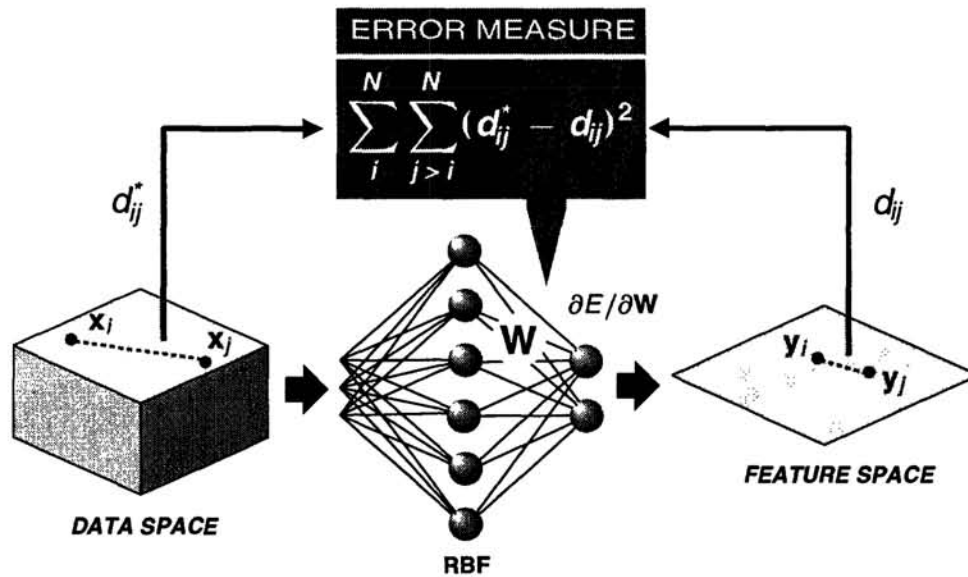

Figure 1: The NEUROSCALE architecture.

## 2   Generalisation

In a supervised learning context, generalisation performance deteriorates for over-complex networks as 'overfitting' occurs. By contrast, it is an interesting empirical observation that the generalisation performance of NEUROSCALE, and related models, is largely insensitive to excessive model complexity. This applies both to the number of centres used in the RBF and in the kernel smoothing factors which themselves may be viewed as regularising hyperparameters in a feed-forward supervised situation.

This insensitivity may be illustrated by Figure 2, which shows the training and test set performances on the IRIS data (for 5-45 basis functions trained and tested on 45 separate samples). To within acceptable deviations, the training and test set

STRESS values are approximately constant. This behaviour is counter-intuitive when compared with research on feed forward networks trained according to supervised approaches. We have observed this general trend on a variety of diverse real world problems, and it is not peculiar to the IRIS data.

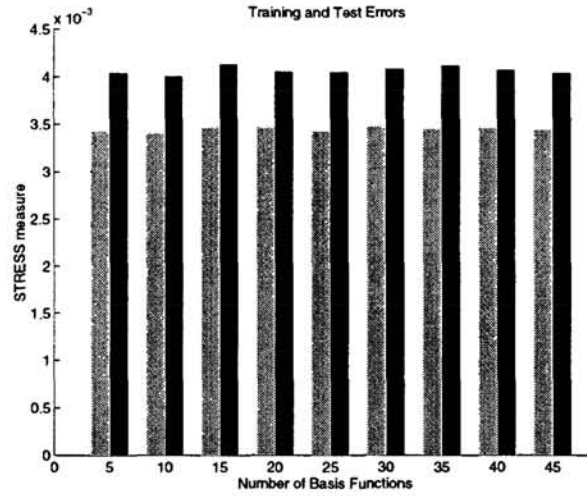

Figure 2: Training and test errors for NEUROSCALE Radial Basis Functions with various numbers of basis functions. Training errors are on the left, test errors are on the right.

There are two fundamental causes of this observed behaviour. Firstly, we may derive significant insight into the necessary form of the functional transformation independent of the data. Secondly, given this prior functional knowledge, there is an appropriate regularising component implicitly incorporated in the training algorithm outlined in the previous section.

## 2.1   Smoothness and Topographic Transformations

For a supervised problem, in the absence of any explicit prior information, the smoothness of the network function must be determined by the data, typically necessitating the setting of regularising hyperparameters to counter overfitting behaviour. In the case of the distance-preserving transformation effected by NEUROSCALE, an understanding of the necessary smoothness may be deduced *a priori*.

Consider a point $\mathbf{x}_q$ in input space and a nearby test point $\mathbf{x}_p = \mathbf{x}_q + \epsilon_{pq}$, where $\epsilon_{pq}$ is an arbitrary displacement vector. Optimum generalisation demands that the distance between the corresponding image points $\mathbf{y}_q$ and $\mathbf{y}_p$ should thus be $\|\epsilon_{pq}\|$. Considering the Taylor expansions around the point $\mathbf{y}_q$ we find

$$\|\mathbf{y}_p - \mathbf{y}_q\|^2 = \sum_{l=1}^{n} (\epsilon_{pq}^{\mathsf{T}} \mathbf{g}_{ql})^2 + O(\epsilon^4),$$

$$= \epsilon_{pq}^{\mathsf{T}} \left( \sum_{l=1}^{n} \mathbf{g}_{ql} \mathbf{g}_{ql}^{T} \right) \epsilon_{pq} + O(\epsilon^4), \tag{3}$$

$$= \epsilon_{pq}^{\mathsf{T}} \mathbf{G}_q \epsilon_{pq} + O(\epsilon^4),$$

where the matrix $\mathbf{G}_q = \sum_{l=1}^{n} \mathbf{g}_{ql}\mathbf{g}_{ql}^{T}$ and $\mathbf{g}_{ql}$ is the gradient vector $(\partial y_l(q)/\partial x_1, \ldots, \partial y_l(q)/\partial x_n)^T$ evaluated at $\mathbf{x} = \mathbf{x}_q$. For structure preservation the corresponding distances in input and output spaces need to be retained for all values of $\epsilon_{pq}$: $\| \mathbf{y}_p - \mathbf{y}_q \|^2 = \epsilon^T \epsilon$, and so $\mathbf{G}_q = \mathbf{I}$ with the requirement that second- and higher-order terms must vanish. In particular note that measures of curvature proportional to $\left(\partial^2 y_l(q)/\partial x_i^2\right)^2$ should vanish. In general, for dimension reduction, we cannot ensure that exact structure preservation is obtained since the rank of $\mathbf{G}_q$ is necessarily less than $n$ and hence can never equate to the identity matrix. However, when minimising STRESS we are locally attempting to minimise the residual $\| \mathbf{I} - \mathbf{G}_q \|$, which is achieved when all the vectors $\epsilon_{pq}$ of interest lie within the range of $\mathbf{G}_q$.

## 2.2 The Training Mechanism

An important feature of this class of topographic transformations is that the STRESS measure is invariant under arbitrary rotations and transformations of the output configuration. The algorithm outlined previously tends towards those configurations that generally reduce the sum-of-squared weight values (Tipping, 1996). This is achieved without any explicit addition of regularisation, but rather it is a feature of the relative supervision algorithm.

The effect of this reduction in weight magnitudes on the smoothness of the network transformation may be observed by monitoring an explicit quantitative measure of *total* curvature:

$$C = \sum_{q}^{N} \sum_{l}^{n} \sum_{i}^{m} \left( \frac{\partial^2 y_l(q)}{\partial x_i^2} \right)^2, \tag{4}$$

where $q$ ranges over the patterns, $i$ over the input dimensions and $l$ over the output dimensions.

Figure 3 depicts the total curvature of NEUROSCALE as a function of the training iterations on the IRIS subset data for a variety of model complexities. As predicted, curvature generally decreases during the training process, with the final value independent of the model complexity. Theoretical insight into this phenomenon is given in (Tipping, 1996).

This behaviour is highly relevant, given the analysis of the previous subsection. That the training algorithm implicitly reduces the sum-of-squares weight values implies that there is a *weight decay* process occurring with an associated smoothing effect. While there is no control over the magnitude of this element, it was shown that for good generalisation, the optimal transformation should be maximally smooth. This self-regularisation operates differently to regularisers normally introduced to stabilise the ill-posed problems of supervised neural network models. In the latter case the regulariser acts to oppose the effect of reducing the error on the training set. In NEUROSCALE the implicit weight decay operates *with* the minimisation of STRESS since the aim is to 'fit' the *relative* input positions exactly.

That there are many RBF networks which satisfy a given STRESS level may be seen by training a network *a posteriori* on a predetermined Sammon mapping of a data set by a supervised approach (since then the targets *are* known explicitly). In general, such *a posteriori* trained networks do not have a low curvature and hence

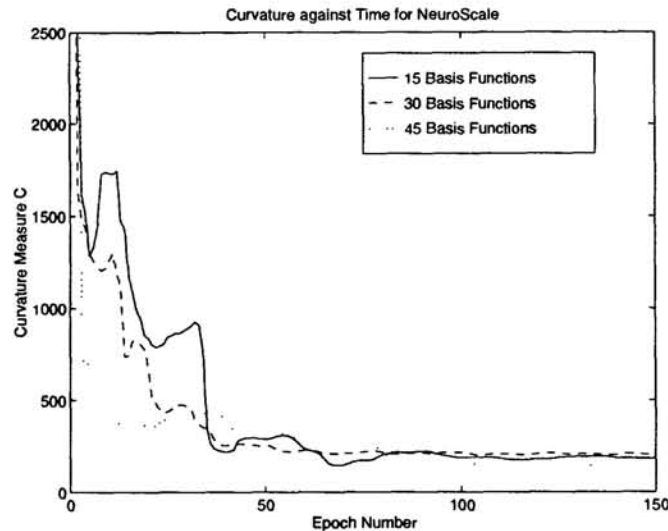

Figure 3: Curvature against time during the training of a NEUROSCALE mapping on the Iris data, for networks with 15, 30 and 45 basis functions.

do not show as good a generalisation behaviour as networks trained according to the relative supervision approach. The method by which NEUROSCALE reduces curvature, is to select, automatically, RBF networks with minimum norm weights. This is an inherent property of the training algorithm to reduce the STRESS criterion.

## 2.3   An example

An effective example of the ease of production of good generalising transformations is given by the following experiment. A synthetic data set comprised four Gaussian clusters, each with spherical variance of 0.5, located in four dimensions with centres at $(x_c, 0, 0, 0) : x_c \in \{1, 2, 3, 4\}$. A NEUROSCALE transformation to two dimensions was trained using the relative supervision approach, using the three clusters at $x_c = 1, 3$ and 4. The network was then tested on the entire dataset, with the fourth cluster included, and the projections are given in Figure 4 below.

The apparently excellent generalisation to test data not sampled from the same distribution as the training data is a function of the inherent smoothing within the training process and also reflects the fact that the test data lay approximately within the range of the matrices $\mathbf{G_q}$ determined during training.

## 3   Conclusion

We have described NEUROSCALE, a parameterised RBF Sammon mapping approach for topographic feature extraction. The NEUROSCALE method may be viewed as a technique which is closely related to Sammon mappings and nonlinear metric MDS, with the added flexibility of producing a generalising transformation.

A theoretical justification has been provided for the empirical observation that the generalisation performance is not affected by model order complexity issues. This counter-intuitive result is based on arguments of necessary transformation smooth-

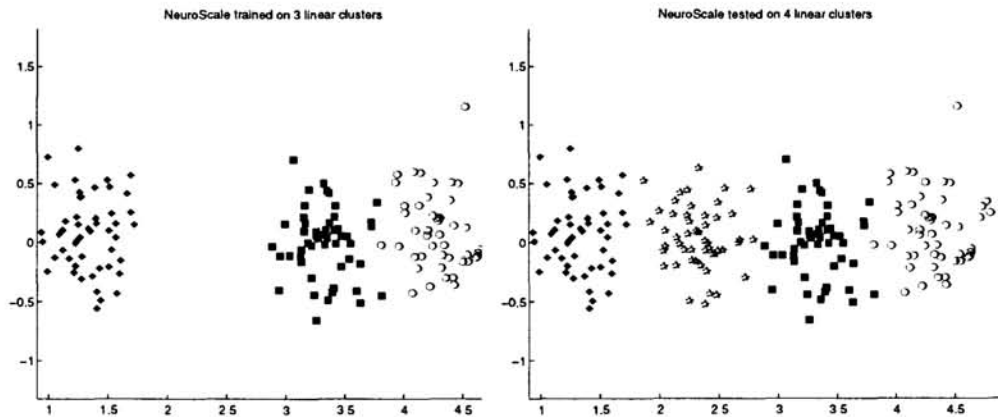

Figure 4: Training and test projections of the four clusters. Training STRESS was 0.00515 and test STRESS 0.00532.

ness coupled with the apparent self-regularising aspects of NEUROSCALE. The relative supervision training algorithm implicitly minimises a measure of curvature by incorporating an automatic 'weight decay' effect which favours solutions generated by networks with small overall weights.

**Acknowledgements**

This work was supported in part under the EPSRC contract GR/J75425, *"Novel Developments in Learning Theory for Neural Networks".*

# References

Kruskal, J. B. (1964). Multidimensional scaling by optimising goodness of fit to a nonmetric hypothesis. *Psychometrika*, 29(1):1–27.

Lowe, D. (1993). Novel 'topographic' nonlinear feature extraction using radial basis functions for concentration coding in the 'artificial nose'. In *3rd IEE International Conference on Artificial Neural Networks*. London: IEE.

Lowe, D. and Tipping, M. E. (1996). Feed-forward neural networks and topographic mappings for exploratory data analysis. *Neural Computing and Applications*, 4:83–95.

Mao, J. and Jain, A. K. (1995). Artificial neural networks for feature extraction and multivariate data projection. *IEEE Transactions on Neural Networks*, 6(2):296–317.

Sammon, J. W. (1969). A nonlinear mapping for data structure analysis. *IEEE Transactions on Computers*, C-18(5):401–409.

Tipping, M. E. (1996). *Topographic Mappings and Feed-Forward Neural Networks*. PhD thesis, Aston University, Aston Street, Birmingham B4 7ET, UK. Available from http://www.ncrg.aston.ac.uk/.

Webb, A. R. (1995). Multidimensional scaling by iterative majorisation using radial basis functions. *Pattern Recognition*, 28(5):753–759.